# On Stochastic and Worst-case Models for Investing

**Elad Hazan**
IBM Almaden Research Center
650 Harry Rd, San Jose, CA 95120
ehazan@cs.princeton.edu

**Satyen Kale**
Yahoo! Research
4301 Great America Parkway, Santa Clara, CA 95054
skale@yahoo-inc.com

## Abstract

In practice, most investing is done assuming a probabilistic model of stock price returns known as the Geometric Brownian Motion (GBM). While often an acceptable approximation, the GBM model is not always valid empirically. This motivates a worst-case approach to investing, called universal portfolio management, where the objective is to maximize wealth relative to the wealth earned by the best fixed portfolio in hindsight.

In this paper we tie the two approaches, and design an investment strategy which is universal in the worst-case, and yet capable of exploiting the mostly valid GBM model. Our method is based on new and improved regret bounds for online convex optimization with exp-concave loss functions.

## 1 Introduction

**"Average-case" Investing:** Much of mathematical finance theory is devoted to the modeling of stock prices and devising investment strategies that maximize wealth gain, minimize risk while doing so, and so on. Typically, this is done by estimating the parameters in a probabilistic model of stock prices. Investment strategies are thus geared to such average case models (in the formal computer science sense), and are naturally susceptible to drastic deviations from the model, as witnessed in the recent stock market crash.

Even so, empirically the Geometric Brownian Motion (GBM) ([Osb59, Bac00]) has enjoyed great predictive success and every year trillions of dollars are traded assuming this model. Black and Scholes [BS73] used this same model in their Nobel prize winning work on pricing options on stocks.

**"Worst-case" Investing:** The fragility of average-case models in the face of rare but dramatic deviations led Cover [Cov91] to take a worst-case approach to investing in stocks. The performance of an online investment algorithm for *arbitrary* sequences of stock price returns is measured with respect to the best CRP (constant rebalanced portfolio, see [Cov91]) in hindsight. A universal portfolio selection algorithm is one that obtains sublinear (in the number of trading periods $T$) *regret*, which is the difference in the logarithms of the final wealths obtained by the two.

Cover [Cov91] gave the first universal portfolio selection algorithm with regret bounded by $O(\log T)$. There has been much follow-up work after Cover's seminal work, such as [HSSW96, MF92, KV03, BK97, HKKA06], which focused on either obtaining alternate universal algorithms or improving the efficiency of Cover's algorithm. However, the best regret bound is still $O(\log T)$.

This dependence of the regret on the number of trading periods is not entirely satisfactory for two main reasons. First, *a priori* it is not clear why the online algorithm should have high regret (growing with the number of iterations) in an unchanging environment. As an extreme example, consider a setting with two stocks where one has an "upward drift" of $1\%$ daily, whereas the second stock remains at the same price. One would expect to "figure out" this pattern quickly and focus on the

first stock, thus attaining a constant fraction of the wealth of the best CRP in the long run, i.e. constant regret, unlike the worst-case bound of $O(\log T)$.

The second problem arises from trading frequency. Suppose we need to invest over a fixed period of time, say a year. Trading more frequently potentially leads to higher wealth gain, by capitalizing on short term stock movements. However, increasing trading frequency increases $T$, and thus one may expect more regret. The problem is actually even worse: since we measure regret as a difference of logarithms of the final wealths, a regret bound of $O(\log T)$ implies a poly$(T)$ factor ratio between the final wealths. In reality, however, experiments [AHKS06] show that some known online algorithms actually *improve* with increasing trading frequency.

**Bridging Worst-case and Average-case Investing:** Both these issues are resolved if one can show that the regret of a "good" online algorithm depends on total *variation* in the sequence of stock returns, rather than purely on the number of iterations. If the stock return sequence has low variation, we expect our algorithm to be able to perform better. If we trade more frequently, then the per iteration variation should go down correspondingly, so the total variation stays the same.

We analyze a portfolio selection algorithm and prove that its regret is bounded by $O(\log Q)$, where $Q$ (formally defined in Section 1.2) is the sum of squared deviations of the returns from their mean. Since $Q \leq T$ (after appropriate normalization), we improve over previous regret bounds and retain the worst-case robustness. Furthermore, in an average-case model such as GBM, the variation can be tied very nicely to the volatility parameter, which explains the experimental observation the regret doesn't increase with increasing trading frequency. Our algorithm is efficient, and its implementation requires constant time per iteration (independent of the number of game iterations).

## 1.1   New Techniques and Comparison to Related Work

Cesa-Bianchi, Mansour and Stoltz [CBMS07] initiated work on relating worst case regret to the variation in the data for the related learning problem of prediction from expert advice, and conjectured that the optimal regret bounds should depend on the observed variation of the cost sequence. Recently, this conjectured was proved and regret bounds of $\tilde{O}(\sqrt{Q})$ were obtained in the full information and bandit linear optimization settings [HK08, HK09], where $Q$ is the variation in the cost sequence. In this paper we give an exponential improvement in regret, viz. $O(\log Q)$, for the case of online *exp-concave* optimization, which includes portfolio selection as a special case.

Another approach to connecting worst-case to average-case investing was taken by Jamshidian [Jam92] and Cross and Barron [CB03]. They considered a model of "continuous trading", where there are $T$ "trading intervals", and in each the online investor chooses a *fixed* portfolio which is rebalanced $k$ times with $k \to \infty$. They prove familiar regret bounds of $O(\log T)$ (independent of $k$) in this model w.r.t. the best fixed portfolio which is rebalanced $T \times k$ times. In this model our algorithm attains the tighter regret bounds of $O(\log Q)$, although our algorithm has more flexibility. Furthermore their algorithms, being extensions of Cover's algorithm, may require exponential time in general[1].

Our bounds of $O(\log Q)$ regret require completely different techniques compared to the $\tilde{O}(\sqrt{Q})$ regret bounds of [HK08, HK09]. These previous bounds are based on first-order gradient descent methods which are too weak to obtain $O(\log Q)$ regret. Instead we have to use the second-order Newton step ideas based on [HKKA06] (in particular, the Hessian of the cost functions).

The second-order techniques of [HKKA06] are, however, not sensitive enough to obtain $O(\log Q)$ bounds. This is because progress was measured in terms of the distance between successive portfolios in the usual Euclidean norm, which is insensitive to variation in the cost sequence. In this paper, we introduce a different analysis technique, based on analyzing the distance between successive predictions using norms that keep changing from iteration to iteration and are actually sensitive to the variation.

A key technical step in the analysis is a lemma (Lemma 6) which bounds the sum of differences of successive Cesaro means of a sequence of vectors by the logarithm of its variation. This lemma,

which may be useful in other contexts when variation bounds on the regret are desired, is proved using the Kahn-Karush-Tucker conditions, and also improves the regret bounds in previous papers.

## 1.2 The model and statement of results

**Portfolio management.** In the universal portfolio management model [Cov91], an online investor iteratively distributes her wealth over $n$ assets before observing the change in asset price. In each iteration $t = 1, 2, \ldots$ the investor commits to an $n$-dimensional distribution of her wealth, $x_t \in \Delta_n = \{\sum_i x_i = 1 \ , \ x \geq 0\}$. She then observes a *price relatives vector* $r_t \in \mathbb{R}^n_+$, where $r_t(i)$ is the ratio between the closing price of the $i^{\text{th}}$ asset on trading period $t$ and the opening price. In the $t^{\text{th}}$ trading period, the wealth of the investor changes by a factor of $(r_t \cdot x_t)$. The overall change in wealth is thus $\prod_t (r_t \cdot x_t)$. Since in a typical market wealth grows at an exponential rate, we measure performance by the exponential growth rate, which is $\log \prod_t (r_t \cdot x_t) = \sum_t \log(r_t \cdot x_t)$. A *constant rebalanced portfolio* (CRP) is an investment strategy which rebalances the wealth in every iteration to keep a fixed distribution. Thus, for a CRP $x \in \Delta_n$, the change in wealth is $\prod_t (r_t \cdot x)$.

The *regret* of the investor is defined to be the difference between the exponential growth rate of her investment strategy and that of the best CRP strategy in hindsight, i.e.

$$\text{Regret} := \max_{x^* \in \Delta_n} \sum_t \log(r_t \cdot x^*) - \sum_t \log(r_t \cdot x_t)$$

Note that the regret doesn't change if we scale all the returns in any particular period by the same amount. So we assume w.l.o.g. that in all periods $t$, $\max_i r_t(i) = 1$. We assume that there is known parameter $r > 0$, such that for all periods $t$, $\min_{t,i} r_t(i) \geq r$. We call $r$ the *market variability* parameter. This is the only restriction we put on the stock price returns; they could be chosen adversarially as long as they respect the market variability bound.

**Online convex optimization.** In the online convex optimization problem [Zin03], which generalizes universal portfolio management, the decision space is a closed, bounded, convex set $K \in \mathbb{R}^n$, and we are sequentially given a series of convex cost[2] functions $f_t : K \to \mathbb{R}$ for $t = 1, 2, \ldots$. The algorithm iteratively produces a point $x_t \in K$ in every round $t$, without knowledge of $f_t$ (but using the past sequence of cost functions), and incurs the cost $f_t(x_t)$. The regret at time $T$ is defined to be

$$\text{Regret} := \sum_{t=1}^{T} f_t(x_t) - \min_{x \in K} \sum_{t=1}^{T} f_t(x).$$

Usually, we will let $\sum_t$ denote $\sum_{t=1}^{T}$. In this paper, we restrict our attention to convex cost functions which can be written as $f_t(x) = g(v_t \cdot x)$ for some univariate convex function $g$ and a parameter vector $v_t \in \mathbb{R}^n$ (for example, in the portfolio management problem, $K = \Delta_n$, $f_t(x) = -\log(r_t \cdot x)$, $g = -\log$, and $v_t = r_t$).

Thus, the cost functions are *parametrized* by the vectors $v_1, v_2, \ldots, v_T$. Our bounds will be expressed as a function of the *quadratic variability* of the parameter vectors $v_1, v_2, \ldots, v_T$, defined as

$$Q(v_1, \ldots, v_T) := \min_\mu \sum_{t=1}^{T} \|v_t - \mu\|^2.$$

This expression is minimized at $\mu = \frac{1}{T} \sum_{t=1}^{T} v_t$, and thus the quadratic variation is just $T - 1$ times the sample variance of the sequence of vectors $\{v_1, \ldots, v_t\}$. Note however that the sequence can be generated adversarially rather than by some stochastic process. We shall refer to this as simply $Q$ if the vectors are clear from the context.

**Main theorem.** In the setup of the online convex optimization problem above, we have the following algorithmic result:

**Theorem 1.** *Let the cost functions be of the form $f_t(x) = g(v_t \cdot x)$. Assume that there are parameters $R, D, a, b > 0$ such that the following conditions hold:*

*1. for all t, $\|v_t\| \leq R$,*
*2. for all $x \in K$, we have $\|x\| \leq D$,*
*3. for all $x \in K$, and for all t, either $g'(v_t \cdot x) \in [0, a]$ or $g'(v_t \cdot x) \in [-a, 0]$, and*
*4. for all $x \in K$, and for all t, $g''(v_t \cdot x) \geq b$.*

*Then there is an algorithm that guarantees the following regret bound:*

$$Regret = O((a^2 n/b) \log(1 + bQ + bR^2) + aRD \log(2 + Q/R^2) + D^2).$$

Now we apply Theorem 1 to the portfolio selection problem. First, we estimate the relevant parameters. We have $\|r_t\| \leq \sqrt{n}$ since all $r_t(i) \leq 1$, thus $R = \sqrt{n}$. For any $x \in \Delta_n$, $\|x\| \leq 1$, so $D = 1$. $g'(v_t \cdot x) = -\frac{1}{(v_t \cdot x)}$, and thus $g'(v_t \cdot x) \in [-\frac{1}{r}, 0]$, so $a = \frac{1}{r}$. Finally, $g''(v_t \cdot x) = \frac{1}{(v_t \cdot x)^2} \geq 1$, so $b = 1$. Applying Theorem 1 we get the following corollary:

**Corollary 2.** *For the portfolio selection problem over $n$ assets, there is an algorithm that attains the following regret bound:*

$$Regret \quad = \quad O\left(\frac{n}{r^2} \log(Q + n)\right).$$

## 2 Bounding the Regret by the Observed Variation in Returns

### 2.1 Preliminaries

All matrices are assumed be real symmetric matrices in $\mathbb{R}^{n \times n}$, where $n$ is the number of stocks. We use the notation $A \succeq B$ to say that $A - B$ is positive semidefinite. We require the notion of a norm of a vector $x$ induced by a positive definite matrix $M$, defined as $\|x\|_M = \sqrt{x^\top M x}$. The following simple generalization of the Cauchy-Schwartz inequality is used in the analysis:

$$\forall x, y \in \mathbb{R}^n : \quad x \cdot y \leq \|x\|_M \|y\|_{M^{-1}}.$$

We denote by $|A|$ the determinant of a matrix $A$, and by $A \bullet B = \mathbf{Tr}(AB) = \sum_{ij} A_{ij} B_{ij}$. As we are concerned with logarithmic regret bounds, potential functions which behave like harmonic series come into play. A generalization of harmonic series to high dimensions is the *vector-harmonic series*, which is a series of quadratic forms that can be expressed as (here $A \succ 0$ is a positive definite matrix, and $v_1, v_2, \ldots$ are vectors in $\mathbb{R}^n$):

$$v_1^\top (A + v_1 v_1^\top)^{-1} v_1, \; v_2^\top (A + v_1 v_1^\top + v_2 v_2^\top)^{-1} v_2, \ldots, v_t^\top (A + \sum_{\tau=1}^t v_\tau v_\tau^\top)^{-1} v_t, \ldots$$

The following lemma is from [HKKA06]:

**Lemma 3.** *For a vector harmonic series given by an initial matrix $A$ and vectors $v_1, v_2, \ldots, v_T$, we have*

$$\sum_{t=1}^T v_t^\top (A + \sum_{\tau=1}^t v_\tau v_\tau^\top)^{-1} v_t \leq \log \left[ \frac{|A + \sum_{\tau=1}^T v_\tau v_\tau^\top|}{|A|} \right].$$

The reader can note that in one dimension, if all vectors $v_t = 1$ and $A = 1$, then the series above reduces exactly to the regular harmonic series whose sum is bounded, of course, by $\log(T + 1)$.

### 2.2 Algorithm and analysis

We analyze the following algorithm and prove that it attains logarithmic regret with respect to the observed variation (rather than number of iterations). The algorithm follows the generic algorithmic scheme of "Follow-The-Regularized-Leader" (FTRL) with squared Euclidean regularization.

---
**Algorithm Exp-Concave-FTL.** In iteration $t$, use the point $x_t$ defined as:

$$x_t \triangleq \arg \min_{x \in \Delta_n} \left( \sum_{\tau=1}^{t-1} f_\tau(x) + \frac{1}{2} \|x\|^2 \right) \qquad (1)$$

---

Note the mathematical program which the algorithm solves is convex, and can be solved in time polynomial in the dimension and number of iterations. The running time, however, for solving this

convex program can be quite high. In the full version of the paper, for the specific problem of portfolio selection, where $f_t(x) = -\log(r_t \cdot x)$, we give a faster implementation whose per iteration running time is independent of the number of iterations, using the more sophisticated "online Newton method" of [HKKA06]. In particular, we have the following result:

**Theorem 4.** *For the portfolio selection problem, there is an algorithm that runs in $O(n^3)$ time per iteration whose regret is bounded by*

$$Regret = O\left(\frac{n}{r^3}\log(Q+n)\right).$$

In this paper, we retain the simpler algorithm and analysis for an easier exposition. We now proceed to prove the Theorem 1.

*Proof.* **[Theorem 1]** First, we note that the algorithm is running a "Follow-the-leader" procedure on the cost functions $f_0, f_1, f_2, \ldots$ where $f_0(x) = \frac{1}{2}\|x\|^2$ is a fictitious period 0 cost function. In other words, in each iteration, it chooses the point that would have minimized the total cost under all the observed functions so far (and, additionally, a fictitious initial cost function $f_0$). This point is referred to as the leader in that round.

The first step in analyzing such an algorithm is to use a stability lemma from [KV05], which bounds the regret of any Follow-the-leader algorithm by the difference in costs (under $f_t$) of the current prediction $x_t$ and the next one $x_{t+1}$, plus an additional error term which comes from the regularization. Thus, we have

$$\text{Regret} \le \sum_t f_t(x_t) - f_t(x_{t+1}) + \frac{1}{2}(\|x^*\|^2 - \|x_0\|^2)$$

$$\le \sum_t \nabla f_t(x_t) \cdot (x_t - x_{t+1}) + \frac{1}{2}D^2$$

$$= \sum_t g'(v_t \cdot x_t)[v_t \cdot (x_t - x_{t+1})] + \frac{1}{2}D^2 \tag{2}$$

The second inequality is because $f_t$ is convex. The last equality follows because $\nabla f_t(x_t) = g'(x_t \cdot v_t)v_t$. Now, we need a handle on $x_t - x_{t+1}$. For this, define $F_t = \sum_{\tau=0}^{t-1} f_\tau$, and note that $x_t$ minimizes $F_t$ over $K$. Consider the difference in the gradients of $F_{t+1}$ evaluated at $x_{t+1}$ and $x_t$:

$$\nabla F_{t+1}(x_{t+1}) - \nabla F_{t+1}(x_t) = \sum_{\tau=0}^{t} \nabla f_\tau(x_{t+1}) - \nabla f_\tau(x_t)$$

$$= \sum_{\tau=1}^{t}[g'_\tau(v_\tau \cdot x_{t+1}) - g'_\tau(v_\tau \cdot x_t)]v_\tau + (x_{t+1} - x_t)$$

$$= \sum_{\tau=1}^{t}[\nabla g'_\tau(v_\tau \cdot \zeta_\tau^t) \cdot (x_{t+1} - x_t)]v_\tau + (x_{t+1} - x_t) \tag{3}$$

$$= \sum_{\tau=1}^{t} g''_\tau(v_\tau \cdot \zeta_\tau^t)v_\tau v_\tau^\top (x_{t+1} - x_t) + (x_{t+1} - x_t). \tag{4}$$

Equation 3 follows by applying the Taylor expansion of the (multi-variate) function $g'_\tau(v_\tau \cdot x)$ at point $x_t$, for some point $\zeta_\tau^t$ on the line segment joining $x_t$ and $x_{t+1}$. The equation (4) follows from the observation that $\nabla g'_\tau(v_\tau \cdot x) = g''_\tau(v_\tau \cdot x)v_\tau$.

Define $A_t = \sum_{\tau=1}^{t} g''(v_t \cdot \zeta_\tau^t)v_t v_t^\top + I$, where $I$ is the identity matrix, and $\Delta x_t = x_{t+1} - x_t$. Then equation (4) can be re-written as:

$$\nabla F_{t+1}(x_{t+1}) - \nabla F_t(x_t) - g'(v_t \cdot x_t)v_t = A_t \Delta x_t. \tag{5}$$

Now, since $x_t$ minimizes the convex function $F_t$ over the convex set $K$, a standard inequality of convex optimization (see [BV04]) states that for any point $y \in K$, we have $\nabla F_t(x_t) \cdot (y - x_t) \ge 0$. Thus, for $y = x_{t+1}$, we get that $\nabla F_t(x_t) \cdot (x_{t+1} - x_t) \ge 0$. Similarly, we get that $\nabla F_{t+1}(x_{t+1}) \cdot (x_t - x_{t+1}) \ge 0$. Putting these two inequalities together, we get that

$$(\nabla F_{t+1}(x_{t+1}) - \nabla F_t(x_t)) \cdot \Delta x_t \le 0. \tag{6}$$

Thus, using the expression for $A_t \Delta x_t$ from (5) we have

$$\|\Delta x_t\|_{A_t}^2 = A_t \Delta x_t \cdot \Delta x_t$$

$$= (\nabla F_{t+1}(x_{t+1}) - \nabla F_t(x_t) - g'(v_t \cdot x_t)v_t) \cdot \Delta x_t$$

$$\le g'(v_t \cdot x_t)[v_t \cdot (x_t - x_{t+1})] \qquad \text{(from (6))} \tag{7}$$

Assume that $g'(v_t \cdot x) \in [-a, 0]$ for all $x \in K$ and all $t$. The other case is handled similarly. Inequality (7) implies that $g'(v_t \cdot x_t)$ and $v_t \cdot (x_t - x_{t+1})$ have the same sign. Thus, we can upper bound

$$g'(v_t \cdot x_t)[v_t \cdot (x_t - x_{t+1})] \leq a(v_t \cdot \Delta x_t). \tag{8}$$

Define $\tilde{v}_t = v_t - \mu_t$, $\mu_t = \frac{1}{t+1}\sum_{\tau=1}^{t} v_\tau$. Then, we have

$$\sum_t v_t \cdot \Delta x_t = \sum_t \tilde{v}_t \cdot \Delta x_t + \sum_{t=2}^{T} x_t(\mu_{t-1} - \mu_t) - x_1\mu_1 + x_{T+1}\mu_T, \tag{9}$$

where $\tilde{v}_t = v_t - \mu_t$, $\mu_t = \frac{1}{t+1}\sum_{\tau=1}^{t} v_t$. Now, define $\rho = \rho(v_1, \ldots, v_T) = \sum_{t=1}^{T-1}\|\mu_{t+1} - \mu_t\|$. Then we bound

$$\sum_{t=2}^{T} x_t(\mu_{t-1} - \mu_t) - x_1\mu_1 + x_{T+1}\mu_T \leq \sum_{t=2}^{T}\|x_t\|\|\mu_{t-1} - \mu_t\| + \|x_1\|\|\mu_1\| + \|x_{T+1}\|\|\mu_T\|$$
$$\leq D\rho + 2DR. \tag{10}$$

We will bound $\rho$ momentarily. For now, we turn to bounding the first term of (9) using the Cauchy-Schwartz generalization as follows:

$$\tilde{v}_t \cdot \Delta x_t \leq \|\tilde{v}_t\|_{A_t^{-1}}\|\Delta x_t\|_{A_t}. \tag{11}$$

By the usual Cauchy-Schwartz inequality,

$$\sum_t \|\tilde{v}_t\|_{A_t^{-1}}\|\Delta x_t\|_{A_t} \leq \sqrt{\sum_t\|\tilde{v}_t\|_{A_t^{-1}}^2} \cdot \sqrt{\sum_t\|\Delta x_t\|_{A_t}^2} \leq \sqrt{\sum_t\|\tilde{v}_t\|_{A_t^{-1}}^2} \cdot \sqrt{\sum_t a(v_t \cdot \Delta x_t)}$$

from (7) and (8). We conclude, using (9), (10) and (11), that

$$\sum_t a(v_t \cdot \Delta x_t) \leq a\sqrt{\sum_t\|\tilde{v}_t\|_{A_t^{-1}}^2} \cdot \sqrt{\sum_t a(v_t \cdot \Delta x_t)} + aD\rho + 2aDR.$$

This implies (using the AM-GM inequality applied to the first term on the RHS) that

$$\sum_t a(v_t \cdot \Delta x_t) \leq a^2\sum_t\|\tilde{v}_t\|_{A_t^{-1}}^2 + 2aD\rho + 4aDR.$$

Plugging this into the regret bound (2) we obtain, via (8),

$$\text{Regret} \leq a^2\sum_t\|\tilde{v}_t\|_{A_t^{-1}}^2 + 2aD\rho + 4aDR + \frac{1}{2}D^2.$$

The proof is completed by the following two lemmas (Lemmas 5 and 6) which bound the RHS. The first term is a vector harmonic series, and the second term can be bounded by a (regular) harmonic series. $\square$

**Lemma 5.** $\sum_t\|\tilde{v}_t\|_{A_t^{-1}}^2 \leq \frac{3n}{b}\log\left[1 + bQ + bR^2\right]$.

*Proof.* We have $A_t = \sum_{\tau=1}^{t} g''(v_t \cdot \zeta_\tau^t)v_t v_t^\top + I$. Since $g''(v_t \cdot \zeta_\tau^t) \geq b$, we have $A_t \succeq I + b\sum_t v_t v_t^\top$. Using the fact that $\tilde{v}_t = v_t - \mu_t$ and $\mu_t = \frac{1}{t+1}\sum_{\tau \leq t} v_\tau$, we get that

$$\sum_{\tau=1}^{t} \tilde{v}_\tau\tilde{v}_\tau^\top = \sum_{s=1}^{t}\left(1 + \sum_{\tau=s}^{t}\frac{1}{(\tau+1)^2}\right)v_s v_s^\top + \sum_{s=1}^{t}\sum_{r<s}\left(-\frac{1}{s} + \sum_{\tau=s}^{t}\frac{1}{(\tau+1)^2}\right)[v_r v_s^\top + v_s v_r^\top].$$

Now, $\sum_{\tau=s}^{t}\frac{1}{(\tau+1)^2} \leq \int_s^{t+1}\frac{1}{x^2}dx = \frac{1}{s} - \frac{1}{t+1}$. Since $(v_r + v_s)(v_r + v_s)^\top \succeq 0$, we get that $v_r v_r^\top + v_s v_s^\top \succeq -[v_r v_s^\top + v_s v_r^\top]$, and hence we have

$$\sum_{\tau=1}^{t} \tilde{v}_\tau\tilde{v}_\tau^\top \preceq \sum_{s=1}^{t}\left(1 + \frac{1}{s}\right)v_s v_s^\top + \sum_{s=1}^{t}\sum_{r<s}\frac{1}{t+1}[v_r v_r^\top + v_s v_s^\top] \preceq \sum_{s=1}^{t}\left(2 + \frac{1}{s}\right)v_s v_s^\top \preceq 3\sum_{s=1}^{t}v_s v_s^\top.$$

Let $\tilde{A}_t = \frac{1}{3}I + b\sum_t \tilde{v}_t\tilde{v}_t^\top$. Note that the inequality above shows that $3\tilde{A}_t \succeq A_t$. Thus, using Lemma 3, we get

$$\sum_t \|\tilde{v}_t\|_{A_t^{-1}}^2 = \sum_t \tilde{v}_t A_t^{-1}\tilde{v}_t \leq \frac{3}{b}\sum_t[\sqrt{b}\tilde{v}_t]^\top \tilde{A}_t^{-1}[\sqrt{b}\tilde{v}_t] \leq \frac{3}{b}\log\left[\frac{|\tilde{A}_T|}{|\tilde{A}_0|}\right]. \tag{12}$$

To bound the latter quantity note that $|\tilde{A}_0| = |I| = 1$, and that

$$|\tilde{A}_T| = |I + b\sum_t\tilde{v}_t\tilde{v}_t^\top| \leq (1 + b\sum_t\|\tilde{v}_t\|_2^2)^n = (1 + b\tilde{Q})^n$$

where $\tilde{Q} = \sum_t \|\tilde{v}_t\|^2 = \sum_t \|\tilde{v}_t - \mu_t\|^2$. Lemma 7 (proved in the full version of the paper), we show that $\tilde{Q} \leq Q + R^2$. This implies that $|\tilde{A}_T| \leq (1 + bQ + bR^2)^n$ and the proof is completed by substituting this bound into (12). $\square$

**Lemma 6.** $\rho(v_1, \ldots, v_T) \leq 2R[\log(2 + Q/R^2) + 1]$.

*Proof.* Define, for $\tau \geq 0$, the vector $u_\tau = v_\tau - \mu_{T+1}$. Note that by convention, we have $v_0 = 0$. We have

$$\sum_{\tau=0}^{T} \|u_\tau\|^2 = \|\mu_{T+1}\|^2 + \sum_{\tau=1}^{T} \|v_\tau - \mu_{T+1}\|^2 = R^2 + Q.$$

Furthermore,

$$\|\mu_{t+1} - \mu_t\| = \left\| \frac{1}{t+2} \sum_{\tau=0}^{t+1} v_\tau - \frac{1}{t+1} \sum_{\tau=0}^{t} v_\tau \right\|$$

$$= \left\| \frac{1}{t+2} \sum_{\tau=0}^{t+1} u_\tau - \frac{1}{t+1} \sum_{\tau=0}^{t} u_\tau \right\|$$

$$\leq \frac{1}{(t+1)^2} \sum_{\tau=0}^{t} \|u_\tau\| + \frac{1}{t+1} \|u_{t+1}\|$$

Summing up over all iterations,

$$\sum_t \|\mu_{t+1} - \mu_t\| \leq \sum_t \left( \frac{1}{(t+1)^2} \sum_{\tau=0}^{t} \|u_\tau\| + \frac{1}{t+1} \|u_{t+1}\| \right) \leq \sum_t \frac{2}{t} \|u_{t-1}\| \leq 2R[\log(2+Q/R^2)+1].$$

The last inequality follows from Lemma 8 (proved in the full version) below by setting $x_t = \|u_{t-1}\|/R$, for $t \geq 1$. $\square$

**Lemma 7.** $\tilde{Q} \leq Q + R^2$.

**Lemma 8.** *Suppose that* $0 \leq x_t \leq 1$ *and* $\sum_t x_t^2 \leq Q$. *Then* $\sum_{t=1}^{T} x_t/t \leq \log(1 + Q) + 1$.

## 3 Implications in the Geometric Brownian Motion Model

We begin with a brief description of the model. The model assumes that stocks can be traded continuously, and that at any time, the fractional change in the stock price within an infinitesimal time interval is normally distributed, with mean and variance proportional to the length of the interval. The randomness is due to many infinitesimal trades that jar the price, much like particles in a physical medium are jarred about by other particles, leading to the classical Brownian motion.

Formally, the model is parameterized by two quantities, the *drift* $\mu$, which is the long term trend of the stock prices, and *volatility* $\sigma$, which characterizes deviations from the long term trend. The parameter $\sigma$ is typically specified as *annualized* volatility, i.e. the standard deviation of the stock's logarithmic returns in one year. Thus, a trading interval of $[0, 1]$ specifies 1 year. The model postulates that the stock price at time $t$, $S_t$, follows a geometric Brownian motion with drift $\mu$ and volatility $\sigma$:

$$dS_t = \mu S_t dt + \sigma S_t dW_t,$$

where $W_t$ is a continuous-time stochastic process known as the Wiener process or simply Brownian motion. The Wiener process is characterized by three facts:

1. $W_0 = 0$,
2. $W_t$ is almost surely continuous, and
3. for any two disjoint time intervals $[s_1, t_1]$ and $[s_2, t_2]$, the random variables $W_{t_1} - W_{s_1}$ and $W_{t_2} - W_{s_2}$ are independent zero mean Gaussian random variables with variance $t_1 - s_1$ and $t_2 - s_2$ respectively.

Using Itō's lemma (see, for example, [KS04]), it can be shown that the stock price at time $t$ is given by

$$S_t = S_0 \exp((\mu - \sigma^2/2)t + \sigma W_t). \tag{13}$$

Now, we consider a situation where we have $n$ stocks in the GBM model. Let $\mu = (\mu_1, \mu_2, \ldots, \mu_n)$ be the vector of drifts, and $\sigma = (\sigma_1, \sigma_2, \ldots, \sigma_n)$ be the vector of (annualized) volatilities. Suppose we trade for one year. We now study the effect of trading frequency on the quadratic variation of the stock price returns. For this, assume that the year-long trading interval is sub-divided into $T$ equally sized intervals of length $1/T$, and we trade at the end of each such interval. Let $r_t = (r_t(1), r_t(2), \ldots, r_t(n))$ be the vector of stock returns in the $t^{\text{th}}$ trading period. We assume that $T$ is "large enough", which is taken to mean that it is larger than $\mu(i), \sigma(i), (\frac{\mu(i)}{\sigma(i)})^2$ for any $i$.

Then using the facts of the Wiener process stated above, we can prove the following lemma, which shows that the expected quadratic variation, and its variance, is the essentially the same regardless of trading frequency. The proof is a straightforward calculation and deferred to the full version of this paper.

**Lemma 9.** *In the setup of trading $n$ stocks in the GBM model over one year with $T$ trading periods, there is a vector $v$ such that*

$$\mathbf{E}\left[\sum_{t=1}^{T}\|r_t - v\|^2\right] \leq \|\sigma\|^2(1 + O(\tfrac{1}{T}))$$

*and*

$$VAR\left[\sum_{t=1}^{T}\|r_t - v\|^2\right] \leq 6\|\sigma\|^2(1 + O(\tfrac{1}{T})),$$

*regardless of how the stocks are correlated.*

Applying this bound in our algorithm, we obtain the following regret bound from Corollary 2.

**Theorem 10.** *In the setup of Lemma 9, for any $\delta > 0$, with probability at least $1 - 2e^{-\delta}$, we have*

$$Regret \leq O(n(\log(\|\sigma\|^2 + n) + \delta)).$$

Theorem 10 shows that one expects to achieve constant regret independent of the trading frequency, as long as the total trading period is fixed. This result is only useful if increasing trading frequency improves the performance of the best constant rebalanced portfolio. Indeed, this has been observed empirically (see e.g. [AHKS06], and more empirical evidence is given in the full version of this paper.).

To obtain a theoretical justification for increasing trading frequency, we consider an example where we have two stocks that follow independent Black-Scholes models with the same drifts, but different volatilities $\sigma_1, \sigma_2$. The same drift assumption is necessary because in the long run, the best CRP is the one that puts all its wealth on the stock with the greater drift. We normalize the drifts to be equal to $0$, this doesn't change the performance in any qualitative manner.

Since the drift is $0$, the expected return of either stock in any trading period is $1$; and since the returns in each period are independent, the expected final change in wealth, which is the product of the returns, is also $1$. Thus, in expectation, any CRP (indeed, any portfolio selection strategy) has overall return $1$. We therefore turn to a different criterion for selecting a CRP. The risk of an investment strategy is measured by the variance of its payoff; thus, if different investment strategies have the same expected payoff, then the one to choose is the one with minimum variance. We therefore choose the CRP with the least variance. We prove the following lemma in the full version of the paper:

**Lemma 11.** *In the setup where we trade two stocks with zero drift and volatilities $\sigma_1, \sigma_2$, the variance of the minimum variance CRP decreases as the trading frequency increases.*

Thus, increasing the trading frequency decreases the variance of the minimum variance CRP, which implies that it gets less risky to trade more frequently; in other words, the more frequently we trade, the more likely the payoff will be close to the expected value. On the other hand, as we show in Theorem 10, the regret does not change even if we trade more often; thus, one expects to see improving performance of our algorithm as the trading frequency increases.

## 4  Conclusions and Future Work

We have presented an efficient algorithm for regret minimization with exp-concave loss functions whose regret strictly improves upon the state of the art. For the problem of portfolio selection, the regret is bounded in terms of the observed variation in stock returns rather than the number of iterations.

Recently, DeMarzo, Kremer and Mansour [DKM06] presented a novel game-theoretic framework for option pricing. Their method prices options using low regret algorithms, and it is possible that our analysis can be applied to options pricing via their method (although that would require a much tighter optimization of the constants involved).

Increasing trading frequency in practice means increasing transaction costs. We have assumed no transaction costs in this paper. It would be very interesting to extend our portfolio selection algorithm to take into account transaction costs as in the work of Blum and Kalai [BK97].

## Footnotes

[1]Cross and Barron give an efficient implementation for some interesting special cases, under assumptions on the variation in returns and bounds on the magnitude of the returns, and assuming $k \to \infty$. A truly efficient implementation of their algorithm can probably be obtained using the techniques of Kalai and Vempala.

[2]Note the difference from the portfolio selection problem: here we have convex *cost* functions, rather than concave payoff functions. The portfolio selection problem is obtained by using $-\log$ as the cost function.

# References

[AHKS06] Amit Agarwal, Elad Hazan, Satyen Kale, and Robert E. Schapire. Algorithms for portfolio management based on the newton method. In *ICML*, pages 9–16, 2006.

[Bac00] L. Bachelier. Théorie de la spéculation. *Annales Scientifiques de l'École Normale Supérieure*, 3(17):21–86, 1900.

[BK97] Avrim Blum and Adam Kalai. Universal portfolios with and without transaction costs. In *COLT*, pages 309–313, New York, NY, USA, 1997. ACM.

[BS73] Fischer Black and Myron Scholes. The pricing of options and corporate liabilities. *Journal of Political Economy*, 81(3):637–654, 1973.

[BV04] Stephen Boyd and Lieven Vandenberghe. *Convex Optimization*. Cambridge University Press, New York, NY, USA, 2004.

[CB03] Jason E Cross and Andrew R Barron. Efficient universal portfolios for past dependent target classes. *Mathematical Finance*, 13(2):245–276, 2003.

[CBMS07] Nicolò Cesa-Bianchi, Yishay Mansour, and Gilles Stoltz. Improved second-order bounds for prediction with expert advice. *Mach. Learn.*, 66(2-3):321–352, 2007.

[Cov91] T. Cover. Universal portfolios. *Math. Finance*, 1:1–19, 1991.

[DKM06] Peter DeMarzo, Ilan Kremer, and Yishay Mansour. Online trading algorithms and robust option pricing. In *STOC '06: Proceedings of the thirty-eighth annual ACM symposium on Theory of computing*, pages 477–486, New York, NY, USA, 2006. ACM.

[HK08] Elad Hazan and Satyen Kale. Extracting certainty from uncertainty: Regret bounded by variation in costs. In *Proceedings of 21st COLT*, 2008.

[HK09] Elad Hazan and Satyen Kale. Better algorithms for benign bandits. In *SODA*, pages 38–47, Philadelphia, PA, USA, 2009. Society for Industrial and Applied Mathematics.

[HKKA06] Elad Hazan, Adam Kalai, Satyen Kale, and Amit Agarwal. Logarithmic regret algorithms for online convex optimization. In *COLT*, pages 499–513, 2006.

[HSSW96] David P. Helmbold, Robert E. Schapire, Yoram Singer, and Manfred K. Warmuth. Online portfolio selection using multiplicative updates. In *ICML*, pages 243–251, 1996.

[Jam92] F. Jamshidian. Asymptotically optimal portfolios. *Mathematical Finance*, 2:131–150, 1992.

[KS04] Ioannis Karatzas and Steven E. Shreve. *Brownian Motion and Stochastic Calculus*. Springer Verlag, New York, NY, USA, 2004.

[KV03] Adam Kalai and Santosh Vempala. Efficient algorithms for universal portfolios. *J. Mach. Learn. Res.*, 3:423–440, 2003.

[KV05] Adam Kalai and Santosh Vempala. Efficient algorithms for online decision problems. *Journal of Computer and System Sciences*, 71(3):291–307, 2005.

[MF92] Neri Merhav and Meir Feder. Universal sequential learning and decision from individual data sequences. In *COLT*, pages 413–427, 1992.

[Osb59] M. F. M. Osborne. Brownian motion in the stock market. *Operations Research*, 2:145–173, 1959.

[Zin03] Martin Zinkevich. Online convex programming and generalized infinitesimal gradient ascent. In *ICML*, pages 928–936, 2003.

